# Learning Group Actions on Latent Representations

**Yinzhu Jin**
Department of Computer Science
University of Virginia
yj3cz@virginia.edu

**Aman Shrivastava**
Department of Computer Science
University of Virginia
as3ek@virginia.edu

**P. Thomas Fletcher**
Department of Electrical and Computer Engineering
Department of Computer Science
University of Virginia
ptf8v@virginia.edu

## Abstract

In this work, we introduce a new approach to model group actions in autoencoders. Diverging from prior research in this domain, we propose to learn the group actions on the latent space rather than strictly on the data space. This adaptation enhances the versatility of our model, enabling it to learn a broader range of scenarios prevalent in the real world, where groups can act on latent factors. Our method allows a wide flexibility in the encoder and decoder architectures and does not require group-specific layers. In addition, we show that our model theoretically serves as a superset of methods that learn group actions on the data space. We test our approach on five image datasets with diverse groups acting on them and demonstrate superior performance to recently proposed methods for modeling group actions.

## 1 Introduction

Group actions are a natural mathematical representation of symmetries and geometric transformations of data. Recent work has demonstrated that explicitly modeling and learning such group actions in neural networks can be beneficial for many tasks, such as learning latent representations [15, 22, 33], generative models [8, 12, 34], and classifiers [1, 4]. While many existing works model group actions on the data space, to the best of our knowledge, nearly all prior works have overlooked group actions on latent factors.

However, there are certain scenarios where we desire to model group actions on factors that are not directly observed. Figure 1 provides an illustrative example. The top row shows a rotating image of a '7', i.e., an orbit under the group action of $SO(2)$ on the image space. (Technically, because of image interpolation, this is only approximately a group action.) This can be effectively modeled using existing approaches designed to learn group actions on the data space. However, if we introduce a slight modification, adding a fixed block, then the digit will be partially occluded for some rotations. The occluded part of the digit is absent from these images but reappears when the digit continues to rotate. This new scenario can no longer be modeled as a group action on the data space. Instead, the group is now acting on the underlying factors. By encoding the representation of these latent factors, specifically the digit in this example, and learning the group action on it, we will be able to correctly model this scenario. While this is a synthesized example, it does exemplify a common phenomenon in real-world images, namely occlusion. Other scenarios in real-world data involve group actions on latent factors. For example, consider taking photographs of a rotating 3D object. The rotation group will be acting on the 3D object geometry but not on the resulting 2D images.

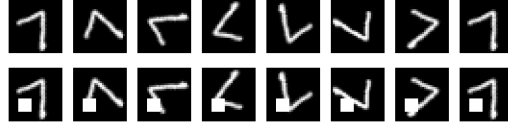

Figure 1: Top row: a group action on the image; bottom row: a group action on the digit but not on the image

We propose to learn group actions in the latent space, without the necessity to conform to group actions on the data space. Employing an autoencoder framework, we learn the latent data representations on which the group acts. Our method is not confined to any specific group or a certain set of groups. It also does not depend on any group-specific layers for encoding and decoding the data, i.e., it allows wide flexibility in the architecture of the autoencoder, as long as it is expressive enough. We present several examples of groups to explain how our model can be applied. Furthermore, we theoretically demonstrate that our model is capable of learning group actions on the data space as well, positioning it as an extension of general data space group action modeling methods.

In summary, there are four key elements to our contributions:

- We propose a method to learn group actions in the latent space, which allows our model to perform group actions on the latent factors.

- Our approach is flexible and can be applied to different groups without group-specific architectures to obtain latent representations. This enables the model to benefit from any advanced deep learning frameworks.

- We focus on rendering new data given the applied group action rather than merely learning the representation.

- The proposed strategy can be seen as an extension of prior works on modeling group actions in the data space. It can still be used to model group actions on the data space, which we show is a special case of the more general setup.

## 2  Related works

There has been considerable research on incorporating group actions into data representation learning and to benefit downstream tasks. For instance, Dey et al. [8] introduces a GAN-type model with a discriminator that is equivariant to the $p4$ or $p4m$ group, resulting in enhanced generative performance even with limited training samples. Moreover, instead of learning the group actions in the data space, Park et al. [21] learn representations equivariant to the latent group actions with a contrastive loss. Wang et al. [30] propose to exploit extrinsic equivariance to model latent representation symmetries and demonstrate benefits in a downstream reinforcement learning task.

Furthermore, part of these works are interested in rendering new data by applying group actions. A significant portion of the studies concentrates on a specific group that acts on the data space. To give an example, Hoogeboom et al. [12] put forth a diffusion model that maintains equivariance to 3D Euclidean transforms at each denoising time step, specifically tailored for 3D molecule generation. In contrast, Yim et al. [34] propose a generative model of protein backbones, modeled as a product of rigid body motions, using a manifold diffusion model [2]. These approaches rely on architectures tailored to be equivariant to a specific group or a set of groups. Cohen and Welling [4] first introduced a group-equivariant convolutional neural network (CNN) featuring group-equivariant convolutional layers. This architecture is designed for groups that represent discrete transformations, including translations, rotations, and reflections. This was followed by multiple works extending group equivariance to broader classes of groups and architectures [5, 23, 31]. In another vein, Satorras et al. [26] present an $E(n)$ equivariant graph neural network, where each graph convolutional layer is inherently designed to be $E(n)$-equivariant.

In alignment with our proposed method, several other works strive to construct models with more flexibility on the types of groups involved. For example, Quessard et al. [22] parameterize a group representation as a product of rotation matrices. The model assumes a finite number of group elements and learns a representation for each. Training involves sequential data collected from a series of

group elements acting on the underlying generative factor in a known order. It is noteworthy that this model uses the product of groups itself as the latent representation. However, a key distinction is that it cannot effectively model group actions on sets containing more than one orbit, due to its direct mapping from the group to the data space. On the other hand, Winter et al. [32] propose learning latent representations consisting of group-invariant components and group elements for acting on the data space. While their framework is not specific to any particular group, the model requires group-specific architectures capable of generating group-invariant representations, e.g., steerable CNNs [5]. In another approach, Hwang et al. [13] suggest predicting the group element that transforms one data point to another. Utilizing a VAE [17] type architecture, they use a Euclidean vector space as the group and define the group action on the data space to correspond to addition in the latent space. This results in a group action on the data space by enforcing the encoder and decoder to both be diffeomorphic mappings. However, this design choice significantly limits the flexibility as it can only model groups that are isomorphic to a real vector space.

Another line of work that is related to our work is novel view synthesis, where the goal is to take an input image of a scene and generate a new image of that scene from a novel camera pose. Although our method is not specifically a novel view synthesis model, i.e., it is a method for more general latent group actions, we do demonstrate its effectiveness in novel view synthesis tasks as an example of 3D rotation group actions. Therefore, we compare to existing geometry-free novel view synthesis methods [9, 28]. Unlike geometry-aware methods [18, 27, 35], geometry-free approaches do not necessitate test-time optimization, which is also the case for our method. For example, Sajjadi et al. [25] utilizes a transformer architecture which also support multiple view inputs. On the other hand, Dupont et al. [9] introduces equivariance into the latent representations. However, unlike our method, they directly apply group actions to the 3D latent volumes as if they are 3D images. This results in higher time complexity despite their relatively small model sizes. Also note that novel view synthesis is by definition intended to learn the view transformation of the whole scene. While our approach is more flexible and for example can be applied to the scenario where only a foreground object is rotating, while the background stays fixed.

## 3 Group actions on latent representations

We will consider an autoencoder that takes a data point $x \in \mathcal{X}$ and encodes it into a latent representation $z \in \mathcal{Z}$ through the encoder mapping $E : \mathcal{X} \to \mathcal{Z}$. The decoder mapping $D : \mathcal{Z} \to \mathcal{X}$ maps a latent representation back into the data space. Typically, the data space, $\mathcal{X}$, and latent space, $\mathcal{Z}$, are real vector spaces, but the mathematical development below does not require this. We will denote the image of $D$ as $\mathcal{X}' = \text{im}(D) \subseteq \mathcal{X}$, which is the space of all possible reconstructions of the decoder. Furthermore, we consider a group $G$ that acts on the latent space $\mathcal{Z}$ via

$$\alpha : G \times \mathcal{Z} \to \mathcal{Z}.$$

When the context is clear, we will denote the group action as $g \,.\, z = \alpha(g, z)$. We will also use the notation $\alpha_g : \mathcal{Z} \to \mathcal{Z}$ to mean $\alpha_g(z) = \alpha(g, z)$. As a reminder, a group action is required to follow two rules for all $g_1, g_2 \in G$ and all $z \in \mathcal{Z}$:

$$e \,.\, z = z,$$
$$g_2 \,.\, (g_1 \,.\, z) = (g_2 g_1) \,.\, z,$$

where $e \in G$ denotes the identity element.

### 3.1 Varying and invariant latent representations

In general scenarios, we need to model both factors that are varying with the group action and those that are invariant to the group action. Coming back to the example of taking photos of rotating 3D objects, the background is invariant to the rotations. Therefore, we propose to split the latent space $\mathcal{Z}$ into a direct product of varying and invariant parts: $\mathcal{Z} = \mathcal{Z}_v \times \mathcal{Z}_i$. We will write a point $z \in \mathcal{Z}$ as

$$z = [z_v; z_i],$$

where $z_v \in \mathcal{Z}_v$ and $z_i \in \mathcal{Z}_i$ represent factors that are varying with and invariant to the group action, respectively. Given a group action $g \,.$ on $\mathcal{Z}_v$, we define the group action on $\mathcal{Z}$ to be:

$$g \,.\, z = [g \,.\, z_v; z_i].$$

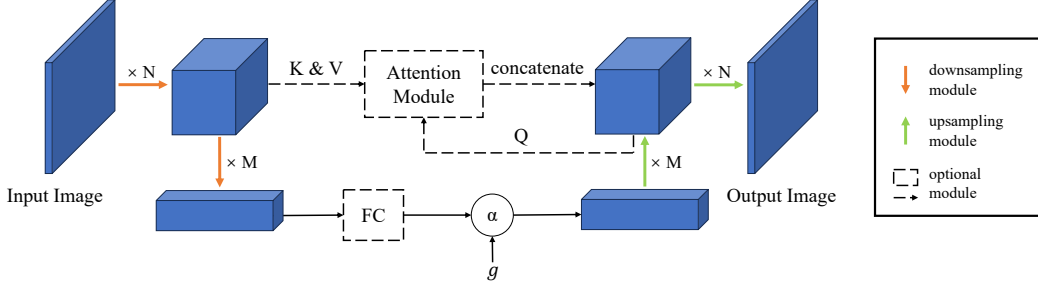

Figure 2: Illustration of our latent space group action model.

It is clear that the identity element $e \in G$ will leave $z$ fixed. Next, we check the associativity:

$$\begin{aligned} g_2 \,.(g_1 \,.\, z) &= [g_2 \,.\, g_1 \,.\, z_v; z_i] \\ &= [(g_2 \cdot g_1) \,.\, z_v; z_i] \\ &= (g_2 \cdot g_1) \,.\, z, \end{aligned}$$

and therefore, this is a valid group action on $\mathcal{Z}$.

### 3.2 Latent space group action model

The goal of training is to jointly learn to autoencode a latent representation of our data along with a group action on those latent factors. For the target group action $\alpha$, we directly compute $\alpha(g, z)$. The output of this group action is then sent as input to a decoder, $D$. During training, we look at a pair of data, $x_1, x_2 \in \mathcal{X}$, at a time, whose corresponding latent representations, $z_1, z_2 \in \mathcal{Z}$, lie in the same orbit, and we are given $g \in G$ such that $z_2 = g \,.\, z_1$.

The loss is simply the reconstruction loss with group actions, where the objective is to minimize:

$$\mathcal{L}_{\mathcal{X}}(x_2, D(g \,.\, z_1)).$$

$\mathcal{L}_{\mathcal{X}}$ can be any standard reconstruction loss appropriate for data in $\mathcal{X}$, e.g., binary cross entropy (BCE) loss or mean square error (MSE), etc. Similarly, we symmetrically add the reconstruction loss of $x_1$, which gives the final form:

$$\mathcal{L}(x_1, x_2) = \mathcal{L}_{\mathcal{X}}(x_2, D(g \,.\, z_1)) + \mathcal{L}_{\mathcal{X}}(x_1, D(g^{-1} \,.\, z_2)).$$

We also find adding LPIPS [36] loss is also helpful for image rendering. It is especially beneficial when the given image only contains very limited information of the latent factor on which the group acts. For example, if we rotate a 3D object, having single random angle of view only gives us scarce information. In our experiments, we found that the perceptual loss guides the model to render more visually convincing images in this circumstance.

### 3.3 Skip connections and attention

When applying group actions to the latent representation $z$, we would like to maintain generalizability by acting on $z$ itself instead of transforming the latent coordinates as done by Dupont et al. [9]. This requires us to sufficiently "mix" spatial dimensions when the given group action is affecting the image globally. However, it is a well-known fact that this will result in some loss of image details [19]. To address this, we include skip connections in our architecture inspired by the U-Net [24].

We skip connect the higher resolution features from the earlier stages of the downsampling path to the corresponding step of the upsampling path. Since the spatial dimensions of the higher resolution features are not mixed adequately, an attention module [29] is applied before concatenating to the upsampling path, where the upsampled feature is acting as the query. This gives us the overall framework of our method, as illustrated in Figure 2. The downsampling module can employ either convolutional layers paired with pooling layers or strided convolutions. Conversely, for upsampling, transposed convolutional layers in conjunction with plain convolutional layers can be utilized.

## 4   Induced group actions on the data space

In this section, we discuss the conditions under which a group action on the latent space, $\mathcal{Z}$, will induce a group action on the output space of the decoder, $\mathcal{X}' = \operatorname{im}(D) \subseteq \mathcal{X}$. It is worth mentioning that we don't define this as a group action on the full data space, $\mathcal{X}$, because it doesn't make sense to consider points in $\mathcal{X}$ that can't be reconstructed by the decoder. We start with a definition:

**Definition 4.1.** A decoder $D$ is called **consistent** with a group action of $G$ on $\mathcal{Z}$ if for any $z_1, z_2 \in \mathcal{Z}$ such that $D(z_1) = D(z_2)$, it is the case that $D(g \,.\, z_1) = D(g \,.\, z_2)$ for any $g \in G$.

**Proposition 4.2.** *Let $D$ be a decoder consistent with a group action $\alpha : G \times \mathcal{Z} \to \mathcal{Z}$. Then $D$ induces a group action $\tilde{\alpha} : G \times \mathcal{X}' \to \mathcal{X}'$ on $\mathcal{X}' = \operatorname{im}(D)$, defined as follows. For any $x \in \mathcal{X}'$, there exists $z \in \mathcal{Z}$ such that $D(z) = x$. Then $\tilde{\alpha}(g, x) = D(g \,.\, z)$. In other words, the following diagram commutes:*

$$
\begin{array}{ccc}
\mathcal{Z} & \xrightarrow{\ \alpha_g\ } & \mathcal{Z} \\
{\scriptstyle D}\downarrow & & \downarrow{\scriptstyle D} \\
\mathcal{X}' & \xrightarrow{\ \tilde{\alpha}_g\ } & \mathcal{X}'
\end{array}
$$

*Proof.* We first note that the induced group action $\tilde{\alpha}$ is a well-defined mapping. That is, $\tilde{\alpha}(g, x)$ does not depend on the latent representation of $x$, precisely because of the consistency condition on $D$. Next, we show that $\tilde{\alpha}$ satisfies the properties of a group action. For any $g_1, g_2 \in G$ and $x \in \mathcal{X}'$, we can pick a $z \in \mathcal{Z}$ such that $x = D(z)$, and we have

$$
\tilde{\alpha}(e, x) = D(\alpha(e, z)) = D(z) = x, \quad \text{and}
$$
$$
\begin{aligned}
\tilde{\alpha}(g_2, \tilde{\alpha}(g_1, x)) &= \tilde{\alpha}(g_2, D(\alpha(g_1, z))) \\
&= D(\alpha(g_2, \alpha(g_2, z))) \\
&= D(\alpha(g_2 g_1, z)) \\
&= \tilde{\alpha}(g_2 g_1, x),
\end{aligned}
$$

which are the two properties for $\tilde{\alpha}$ to be a group action. $\qquad\square$

The consistency condition of Definition 4.1 is difficult to check directly. The next result shows a more intuitive condition that a decoder that can be inverted by an encoder satisfies the consistency condition for any group action on its latent space.

**Proposition 4.3.** *Let $E : \mathcal{X} \to \mathcal{Z}$ and $D : \mathcal{Z} \to \mathcal{X}$ be an autoencoder such that $E(D(z)) = z$ for all $z \in \mathcal{Z}$. Let $\alpha : G \times \mathcal{Z} \to \mathcal{Z}$ be a group action. Then $D$ is consistent with $\alpha$ and thus induces a group action $\tilde{\alpha}$ on $\mathcal{X}'$ (as defined in Proposition 4.2).*

*Proof.* The condition $E(D(z)) = z$ implies that $D$ is injective. To see this, consider two points $z_1, z_2 \in \mathcal{Z}$ such that $z_1 \neq z_2$. Then it must be the case that $D(z_1) \neq D(z_2)$ in order for $E(D(z_1)) = z_1 \neq z_2 = E(D(z_2))$. Next, from Definition 4.1 it is clear that injectivity of $D$ implies it is consistent with any group action. $\qquad\square$

We note that the condition that $E(D(z)) = z$ for all $z \in \mathcal{Z}$, i.e., that $E$ be a left-inverse of $D$, does not imply $D(E(x)) = x$. In other words, the encoder and decoder are not necessarily (two-sided) inverses of each other, as is the case in [13]. In that work, the two-sided inverse requirement means that $\mathcal{Z}$ and $\mathcal{X}$ must be vector spaces of the same dimension. For the one-sided inverse condition in Proposition 4.3, $\mathcal{Z}$ and $\mathcal{X}$ do not have to be isomorphic sets. For example, if $\mathcal{Z}$ and $\mathcal{X}$ are vector spaces, $\mathcal{Z}$ can have smaller dimension than $\mathcal{X}$.

## 5   Examples

### 5.1   2D and 3D rotations

We look at 2D and 3D rotation groups, $\operatorname{SO}(2)$ and $\operatorname{SO}(3)$. The rotation matrix $g \in \operatorname{SO}(k)$ acts on $\mathcal{Z}$ in the manner described in Section 3.1. As for the group action of $\operatorname{SO}(k)$ on $\mathcal{Z}_v$, we apply a given

rotation matrix to every subset of $k$ dimensions separately. More specifically, we reshape $z_v$ into an $k \times n$ matrix, with $n = \dim(z_v)/k$. Denoting this matrix as $z'_v$, we define the group action on $z$ as:

$$g \cdot z = [\text{vec}(g \cdot z'_v); z_i],$$

where $\text{vec}()$ indicates reshaping the given matrix into a vector.

## 5.2 Image contrast transformations

An image contrast transformation is a diffeomorphic function from $[0,1]$ to $[0,1]$ that is applied to each pixel value. We can define a two-parameter family of image contrast transformation as the affine group on the real line, $G = \text{Aff}(\mathbb{R}^1) = \mathbb{R}_{>0} \ltimes \mathbb{R}$, as follows. Note an element $(a,b) \in G$ represents a scaling by $a$ and shift by $b$ of the real line. The group operation of $G$ is $(a_1, b_1) \cdot (a_2, b_2) = (a_1 a_2, b_1 + a_1 b_2)$. Now, we show that $G$ can also act on the unit interval. Let $(a,b) \in G$ and $x \in [0,1]$ be a pixel intensity, and define

$$(a,b) \cdot x = \sigma(a \log it(x) + b).$$

The sigmoid function $\sigma$ and $\log it$ are inverse functions of each other. We can prove this is a valid group action on $[0,1]$, where the group being $\text{Aff}(1)$. The output of $\sigma()$ is always in $[0,1]$, and therefore, $[0,1]$ is closed under the group action. We can model this in the latent space with a simpler $\text{Aff}(\mathbb{R}^1)$ group action:

$$(a,b) \cdot z = [a z_v + b; z_i].$$

## 5.3 Cyclic group transformations

In additional to the continuous groups, our method can also be applied to discrete group actions. The cyclic group $C_k$ is equivalent to the set $\{0, 1, 2, \dots, k-1\}$ equipped with the binary operation defined as addition modulo $k$. In implementation, this can be represented as the subset of $\text{SO}(2)$ consisting of discrete rotations by angles $\{0, \frac{2\pi}{k}, \frac{4\pi}{k}, \dots, \frac{2\pi(k-1)}{k}\}$. Then this representation of $C_k$ can be incorporated into our model as described in Section 5.1.

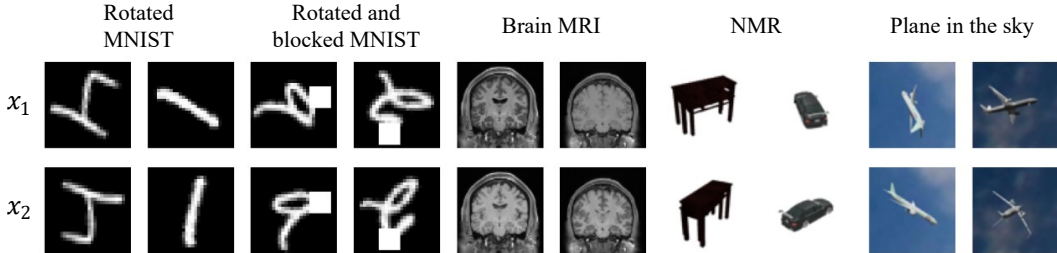

Figure 3: Data samples used in the experiments. Each column represents a pair related by a group action, i.e., $x_1 = g \cdot x_2$.

# 6 Experiments

We conduct experiments on five different image datasets. Our models and all the baseline models train on pairs of images. We show some sample pairs from each dataset in Figure 3.

## 6.1 Datasets

**Rotated MNIST** dataset is obtained by simply rotating images from MNIST dataset [6] about the center by random angles $\theta$, where $\theta \sim \mathcal{U}[0, 2\pi]$. Bilinear interpolation is used for sampling. We use the original train-test split and image size of $28 \times 28$. The group action for this dataset is 2D rotation with the group being $\text{SO}(2)$. The ground truth actions between each pair is easily computable from the rotation angles. To ensure that the group action is also on the data space, as stated in Proposition 4.3, we introduce an additional loss term, $\|z - E(D(z))\|_2^2$, where $z$ denotes latents encoded from real data. Please see the appendix for more details.

**Rotated and blocked MNIST** dataset is further processed from the rotated MNIST dataset by adding one randomly placed $7 \times 7$ white square to each image with probability $0.8$. The group action is still

2D rotation. Moreover, images of a pair either have no squares or have squares at the same locations. In other words, the square is invariant to the group action. This gives us a dataset where the rotation is acted on the latent factor—representation of the digits—and not on the whole image.

**Brain MRI** dataset is derived from the Human Connectome Project [11]. It consists of 1113 3D brain magnetic resonance images with brain segmentations from FreeSurfer [10]. We took mid-coronal 2D slices for each subject, among which 880 images are used for training and 233 for testing. Then the image contrast transformation defined in Section 5.2 is performed only to the pixels in the brain, leaving the remaining parts (skull, neck, etc.) unchanged. Note that this is not a simple contrast transformation on the whole image. The model needs to identify the brain pixels, and thus, this is again a group action on a latent factor and not the data itself. We randomly sample $a = e^t, t \sim \mathcal{N}(0, 0.25)$ and $b \sim \mathcal{N}(0, 0.25)$ 100 times for each original image.

**Neural 3D mesh renderer (NMR)** [14] dataset has been used in multiple previous works in the field of novel view synthesis. This dataset is derived from ShapeNet [3] by rendering each object at 24 fixed views around the object in a cycle. This forms a cyclic group, $C_{24}$, acting on the latent factor - the camera angle. We stick to the original split for training and evaluations.

**Plane in the sky** dataset is our own rendering of ShapeNet Core [3] airplane objects. Each airplane is put in a real sky background cropped from the SWIMSEG sky image dataset [7]. We uniformly sample 100 random 3D rotation matrices from $SO(3)$ and rotate the plane. Different from the novel view synthesis problem, this results in sets of images with the same sky background and varying plane orientations. We randomly split out 20% as the testing set.

## 6.2    Comparison to baselines

**Implementation details and baseline models.**    All our experiments are implemented with the PyTorch [20] package. As illustrated in Figure 2, our encoder consists of several convolutional downsampling modules, while our decoder comprises convolutional upsampling modules. We used skip connections and attention modules for the two rendered 3D objects datasets. For more details of model architectures for each dataset, readers can refer to the appendix. The training is performed by randomly sampling pairs of data points from the same orbit. We used MSE for the reconstruction loss, and further added LPIPS loss for rendered 3D objects datasets with a weight of 0.0005.

For the two MNIST derived datasets, we compare to Hwang et al. [13] and Winter et al. [32]. While for the brain MRI set, we only compare to Hwang et al. [13], since Winter et al. [32] requires tailored equivariant layer for each group and there is no readily available one for the image contrast transformation group. For the two datasets that are rendered images of 3D objects, we compare to two novel view synthesis models - Sajjadi et al. [25] and Dupont et al. [9].

For the encoders of Hwang et al. [13], which encode the group action given each pair, we use almost identical architectures as our encoders. The only difference is that their encoders are variational and require reparameterization. We carefully follow their work to implement the decoders, which are built upon Glow [16] type of normalizing flow architectures. All the other baselines have official or officially recognized implementations.

To evaluate the model's ability in learning group actions accurately, we compare the predicted $D(g \cdot z)$ with the ground truth image. Since Hwang et al. [13] encodes group actions based on pairs only, we present another pair of samples, $x_3$ and $x_4$, s.t. $x_4 = g \cdot x_3$, for their models to encode $g$. In our experiments, we find their model performs best when given $x_3$ and $x_4$ having the same absolute angles, or the same $(a, b)$ w.r.t. the original images, as $x_1$ and $x_2$ respectively.

**Qualitative results.**    The sample reconstructions for the first three datasets are shown in Figure 4. As we can see, Hwang et al. [13] demonstrates problems learning correct rotation angles for both MNIST derived datasets.. Visually, Winter et al. [32] and our models exhibits comparable performance on the rotated MNIST, indicating the capability of our models to model group actions on $\mathcal{X}$ as well. As for the rotated and blocked MNIST, their model attempts to learn rotation invariant representations, leading to challenges in correctly modeling the blocks. At times, the model tends to align blocks with the digit structures, as the rotation is acted on the digit, as illustrated in the first column of the sample outputs. While for the brain MRI dataset, both models show impressive performance, and it is difficult to determine a noticeable advantage visually.

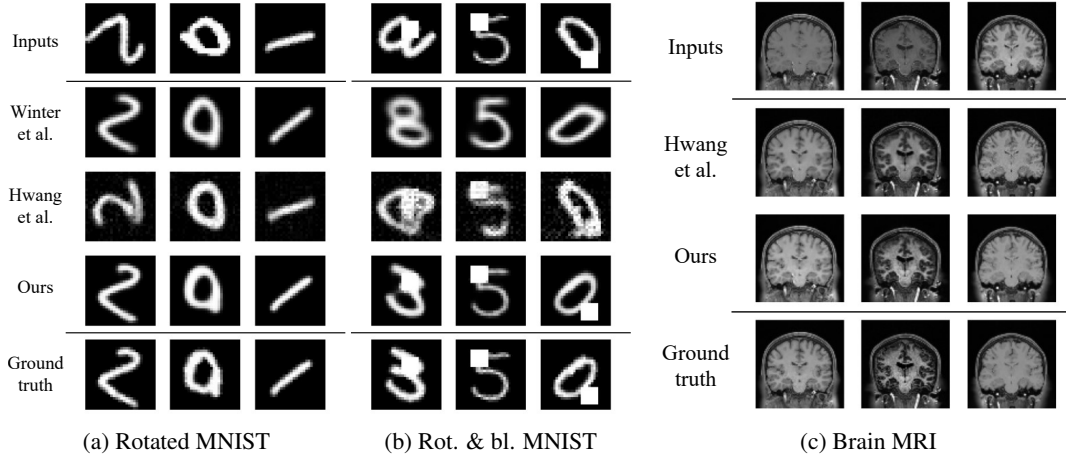

(a) Rotated MNIST    (b) Rot. & bl. MNIST    (c) Brain MRI

Figure 4: Sample reconstructions on MNIST derived datasets and brain MRI dataset

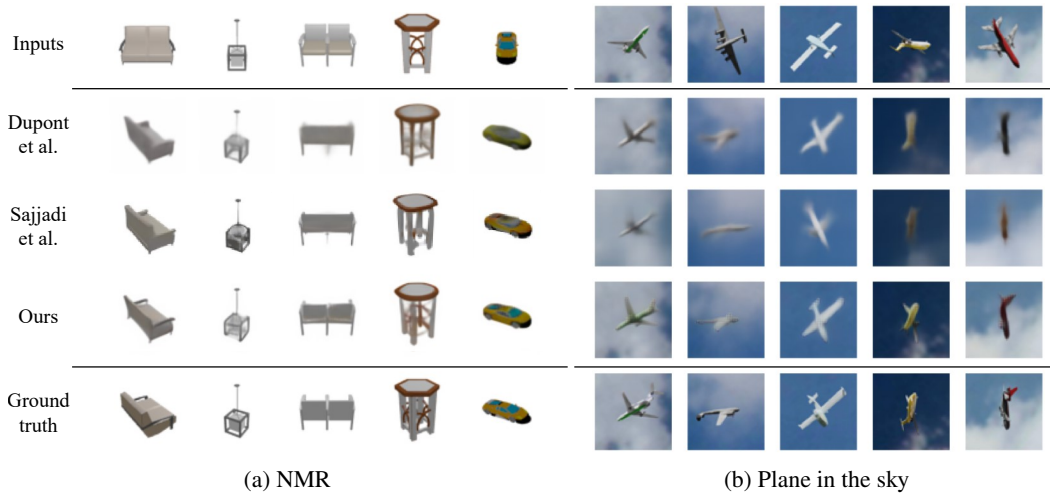

(a) NMR    (b) Plane in the sky

Figure 5: Sample reconstructions on NMR dataset and plane in the sky dataset

The sample reconstructions for the two rendered 3D objects datasets are shown in Figure 5. As NMR is a novel view synthesis and two baseline models are designed for this task, we can see they both render reasonable reconstructions with Sajjadi et al. [25] giving sharper images. Our model also correctly learns view angle changes and even captures image details better in some cases. While for the plane in the sky dataset, which is not a novel view synthesis dataset, we can see both baselines struggle to give convincing results. Dupont et al. [9] has relatively better reconstructions while Sajjadi et al. [25] sometimes miss the plane orientation significantly. Our model renders much higher quality images with correct orientations and better details.

Table 1: Quantitative results on MNIST derived datasets and brain MRI dataset

| | Rotated MNIST | | Rot. & bl. MNIST | | Brain MRI | |
|---|---|---|---|---|---|---|
| | ↑PSNR | ↑SSIM | ↑PSNR | ↑SSIM | ↑PSNR | ↑SSIM |
| Winter et al. [32] | 21.97 | 0.874 | 14.05 | 0.586 | NA | NA |
| Hwang et al. [13] | 15.29 | 0.992 | 10.19 | 0.990 | 27.43 | **1.000** |
| **Ours** | **26.07** | **1.000** | **23.55** | **1.000** | **35.99** | **1.000** |

Table 2: Quantitative results on 3D objects rendered datasets

| | NMR | | | Plane in the sky | | |
|---|---|---|---|---|---|---|
| | ↑PSNR | ↑SSIM | ↓LPIPS | ↑PSNR | ↑SSIM | ↓LPIPS |
| Dupont et al. [9] | 26.91 | 0.899 | 0.091 | 24.25 | 0.773 | 0.239 |
| Sajjadi et al. [25] | 27.87 | 0.912 | 0.066 | 23.53 | 0.489 | 0.280 |
| **Ours** | **28.91** | **0.947** | **0.050** | **25.24** | **0.821** | **0.112** |

**Quantitative results.** We chose peak signal-to-noise ratio (PSNR) and structural similarity index measure (SSIM) as quantitative metrics for the first three datasets and present resulting values in Table 1. Our models consistently achieved the best results across all three datasets and both metrics. For the NMR dataset and plane in the sky dataset, we further compute LPIPS (VGG) in addition to PSNR and SSIM. The results are reported in Table 2. Our models also achieves best results consistently, which aligns with the qualitative results. We can also observe that between the two baselines, Sajjadi et al. [25] is performing better on NMR dataset, while Dupont et al. [9] performing better on the plane in the sky dataset. Finally, it's also worth noting that we checked how well Proposition 4.3 is satisfied with the additional training loss to encourage it. The average $z$ reconstruction $L2$ distance is $0.304$. Comparing to the standard deviation of $z$ over the dataset being $4.043$, we conclude that this property is approximately met.

**Discussion.** It is worth noting that the model by Winter et al. [32] is specifically tailored for group actions on the data space $\mathcal{X}$, which accounts for the performance discrepancy on the rotated and blocked MNIST, where the block does not rotate with the rest of the image. Additionally, even the rotated MNIST is not precisely a group action on $\mathcal{X}$, given the interpolations occurring during rotation. Their original experiment utilized blurred rotated MNIST, different from our dataset, as a measure to mitigate this problem. As for Hwang et al. [13], it models the group as a Euclidean space $\mathbb{R}^d$ under addition in the latent space. We can see that their model especially struggles on the rotation groups, $SO(2)$, which are not Euclidean spaces. Furthermore, the normalizing flow architecture used for encoder/decoder enforces these mappings to be diffeomorphisms, resulting in a group action on $\mathcal{X}$, which accounts for its poor performance on the blocked MNIST.

Dupont et al. [9] incorporates group equivariance in the latent space by transforming the latent coordinates. Although this exactly models novel view synthesis task, it is too rigid for more general scenarios, as in the plane in the sky dataset. Furthermore, we can see that it is not very good at capturing details from the results on NMR dataset. However, the introduction of equivariance might have helped modeling more difficult 3D rotations of the planes, giving it better performance compared to Sajjadi et al. [25]. On the other hand, Sajjadi et al. [25] is a transformer type of model that does not incorporate group equivariance. It is better at capturing details and rendering sharp images as seen in results on NMR dataset. However, it fails on the plane dataset, which is a more challenging task than most novel view synthesis datasets, given full 3D random rotations and single input view.

Table 3: Quantitative results of ablation study on NMR dataset

| | ↑PSNR | ↑SSIM | ↓LPIPS |
|---|---|---|---|
| Ablation 1 | 27.31 | 0.933 | 0.080 |
| Ablation 2 | 26.96 | 0.930 | 0.063 |
| Ablation 3 | 28.09 | 0.941 | 0.072 |
| **Ours** (full) | **28.91** | **0.947** | **0.050** |

## 6.3 Ablation study

We also performed an ablation study on the NMR dataset to explore how skip connections paired with attention modules and the addition of LPIPS loss impact our model performance. We define the following three ablation models: Ablation 1 – our model without the skip connection and trained without the LPIPS loss; Ablation 2 – our model without the skip connection; Ablation 3 – our model trained without the LPIPS loss. The results are listed in Table 3 and some samples are shown in Figure 6. Combining skip connections with LPIPS loss yields the best performance across all metrics,

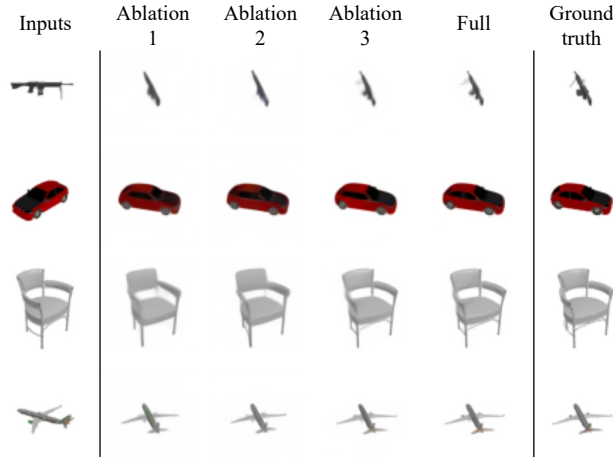

| Inputs | Ablation 1 | Ablation 2 | Ablation 3 | Full | Ground truth |

Figure 6: Samples from ablation models and our full model.

significantly enhancing image details. Skip connections alone provide a performance boost, but adding LPIPS loss without skip connections offers minimal improvement. Notably, even without skip connections or LPIPS, our model accurately captures object orientation and performs comparably to baseline models, highlighting the robustness of our latent group action modeling strategy.

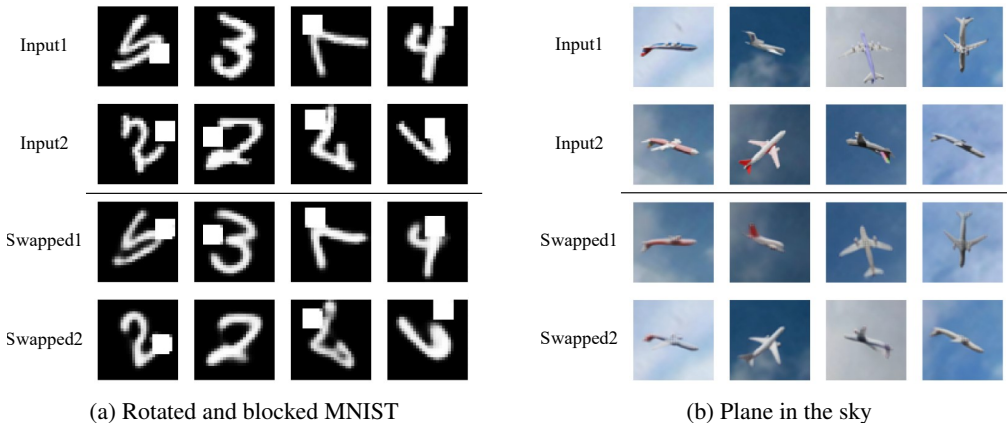

(a) Rotated and blocked MNIST

(b) Plane in the sky

Figure 7: Samples generated by swapping invariant and varying parts of latent representations.

We further investigate how varying and invariant information is captured by swapping the $z_v$ and $z_i$ components of the latent representations between two inputs and decoding the results. Some examples are shown in Figure 7. Interestingly, the model generalizes to new combinations of varying factors (e.g., digit and plane shapes) and invariant factors (e.g., the white block and the sky), despite not being explicitly trained in such fashion.

## 7   Conclusion

In this paper, we propose a novel approach of learning group actions on latent representations. Our experimental results demonstrate that our method can effectively model a broader range of scenarios than existing models of group actions on the data space. In addition to being able to model group actions in the latent space, we show both theoretically and empirically that our strategy is also capable of modeling group actions on the data space. Furthermore, we achieve state of the art performance on the geometry-free novel view synthesis task, and we outperform previous approaches to learning group actions in more general cases. We note that our model requires ground truth group actions during training, which might not be available in some cases. We leave it as future work to apply our method in semi-supervised or unsupervised manners.

## Acknowledgments and Disclosure of Funding

This work was partially supported by NSF Smart and Connected Health grant 2205417.

## References

[1] F. Ahmed, Y. Bengio, H. van Seijen, and A. C. Courville. Systematic generalisation with group invariant predictions. In *9th International Conference on Learning Representations, ICLR*, 2021. 1

[2] V. D. Bortoli, E. Mathieu, M. J. Hutchinson, J. Thornton, Y. W. Teh, and A. Doucet. Riemannian score-based generative modelling. In *Advances in Neural Information Processing Systems 35: Annual Conference on Neural Information Processing Systems 2022, NeurIPS*, 2022. 2

[3] A. X. Chang, T. Funkhouser, L. Guibas, P. Hanrahan, Q. Huang, Z. Li, S. Savarese, M. Savva, S. Song, H. Su, J. Xiao, L. Yi, and F. Yu. ShapeNet: An Information-Rich 3D Model Repository. Technical Report arXiv:1512.03012 [cs.GR], Stanford University — Princeton University — Toyota Technological Institute at Chicago, 2015. 7

[4] T. Cohen and M. Welling. Group equivariant convolutional networks. In *Proceedings of the 33nd International Conference on Machine Learning, ICML*, volume 48, pages 2990–2999, 2016. 1, 2

[5] T. S. Cohen and M. Welling. Steerable CNNs. In *5th International Conference on Learning Representations, ICLR*, 2017. 2, 3

[6] L. Deng. The MNIST database of handwritten digit images for machine learning research. *IEEE Signal Processing Magazine*, 29(6):141–142, 2012. 6

[7] S. Dev, Y. H. Lee, and S. Winkler. Color-based segmentation of sky/cloud images from ground-based cameras. *IEEE Journal of Selected Topics in Applied Earth Observations and Remote Sensing*, 10(1):231–242, 2016. 7

[8] N. Dey, A. Chen, and S. Ghafurian. Group equivariant generative adversarial networks. In *9th International Conference on Learning Representations, ICLR*, 2021. 1, 2

[9] E. Dupont, M. B. Martin, A. Colburn, A. Sankar, J. Susskind, and Q. Shan. Equivariant neural rendering. In *International Conference on Machine Learning*, pages 2761–2770. PMLR, 2020. 3, 4, 7, 8, 9, 17

[10] B. Fischl, D. H. Salat, E. Busa, M. Albert, M. Dieterich, C. Haselgrove, A. Van Der Kouwe, R. Killiany, D. Kennedy, S. Klaveness, et al. Whole brain segmentation: automated labeling of neuroanatomical structures in the human brain. *Neuron*, 33(3):341–355, 2002. 7

[11] M. F. Glasser, S. N. Sotiropoulos, J. A. Wilson, T. S. Coalson, B. Fischl, J. L. Andersson, J. Xu, S. Jbabdi, M. Webster, J. R. Polimeni, et al. The minimal preprocessing pipelines for the human connectome project. *Neuroimage*, 80:105–124, 2013. 7

[12] E. Hoogeboom, V. G. Satorras, C. Vignac, and M. Welling. Equivariant diffusion for molecule generation in 3D. In *International Conference on Machine Learning, ICML*, volume 162, pages 8867–8887, 2022. 1, 2

[13] G. Hwang, J. Choi, H. Cho, and M. Kang. Maganet: Achieving combinatorial generalization by modeling a group action. In *International Conference on Machine Learning, ICML*, volume 202, pages 14237–14248, 2023. 3, 5, 7, 8, 9, 17

[14] H. Kato, Y. Ushiku, and T. Harada. Neural 3d mesh renderer. In *Proceedings of the IEEE conference on computer vision and pattern recognition*, pages 3907–3916, 2018. 7

[15] H. Keurti, H.-R. Pan, M. Besserve, B. F. Grewe, and B. Schölkopf. Homomorphism AutoEncoder – learning group structured representations from observed transitions. In *International Conference on Machine Learning, ICML*, volume 202, pages 16190–16215, 2023. 1

[16] D. P. Kingma and P. Dhariwal. Glow: Generative flow with invertible 1x1 convolutions. In *Advances in Neural Information Processing Systems 31: Annual Conference on Neural Information Processing Systems, NeurIPS*, pages 10236–10245, 2018. 7

[17] D. P. Kingma and M. Welling. Auto-encoding variational Bayes. In *2nd International Conference on Learning Representations, ICLR*, 2014. 3

[18] K.-E. Lin, Y.-C. Lin, W.-S. Lai, T.-Y. Lin, Y.-C. Shih, and R. Ramamoorthi. Vision transformer for nerf-based view synthesis from a single input image. In *Proceedings of the IEEE/CVF Winter Conference on Applications of Computer Vision*, pages 806–815, 2023. 3

[19] J. Long, E. Shelhamer, and T. Darrell. Fully convolutional networks for semantic segmentation. In *Proceedings of the IEEE conference on computer vision and pattern recognition*, pages 3431–3440, 2015. 4

[20] A. Oring. *Autoencoder image interpolation by shaping the latent space*. PhD thesis, Reichman University (Israel), 2021. 7

[21] J. Y. Park, O. Biza, L. Zhao, J.-W. Van De Meent, and R. Walters. Learning symmetric embeddings for equivariant world models. In *International Conference on Machine Learning*, pages 17372–17389. PMLR, 2022. 2

[22] R. Quessard, T. D. Barrett, and W. R. Clements. Learning disentangled representations and group structure of dynamical environments. In *Advances in Neural Information Processing Systems 33: Annual Conference on Neural Information Processing Systems 2020, NeurIPS*, 2020. 1, 2

[23] D. Romero, E. Bekkers, J. Tomczak, and M. Hoogendoorn. Attentive group equivariant convolutional networks. In *Proceedings of the 37th International Conference on Machine Learning, ICML*, pages 8188–8199, 2020. 2

[24] O. Ronneberger, P. Fischer, and T. Brox. U-net: Convolutional networks for biomedical image segmentation. In *Medical image computing and computer-assisted intervention–MICCAI 2015: 18th international conference, Munich, Germany, October 5-9, 2015, proceedings, part III 18*, pages 234–241. Springer, 2015. 4

[25] M. S. Sajjadi, H. Meyer, E. Pot, U. Bergmann, K. Greff, N. Radwan, S. Vora, M. Lučić, D. Duckworth, A. Dosovitskiy, et al. Scene representation transformer: Geometry-free novel view synthesis through set-latent scene representations. In *Proceedings of the IEEE/CVF Conference on Computer Vision and Pattern Recognition*, pages 6229–6238, 2022. 3, 7, 8, 9, 17

[26] V. G. Satorras, E. Hoogeboom, and M. Welling. E(n) equivariant graph neural networks. In *Proceedings of the 38th International Conference on Machine Learning, ICML*, volume 139, pages 9323–9332, 2021. 2

[27] V. Sitzmann, M. Zollhöfer, and G. Wetzstein. Scene representation networks: Continuous 3d-structure-aware neural scene representations. In *Advances in Neural Information Processing Systems 32: Annual Conference on Neural Information Processing Systems, NeurIPS*, pages 1119–1130, 2019. 3

[28] V. Sitzmann, S. Rezchikov, B. Freeman, J. Tenenbaum, and F. Durand. Light field networks: Neural scene representations with single-evaluation rendering. In *Advances in Neural Information Processing Systems*, volume 34, pages 19313–19325, 2021. 3

[29] A. Vaswani, N. Shazeer, N. Parmar, J. Uszkoreit, L. Jones, A. N. Gomez, Ł. Kaiser, and I. Polosukhin. Attention is all you need. *Advances in neural information processing systems*, 30, 2017. 4, 16

[30] D. Wang, J. Y. Park, N. Sortur, L. Wong, R. Walters, and R. Platt. The surprising effectiveness of equivariant models in domains with latent symmetry. In *NeurIPS 2023 Workshop on Symmetry and Geometry in Neural Representations*, 2023. 2

[31] M. Weiler, M. Geiger, M. Welling, W. Boomsma, and T. Cohen. 3D steerable CNNs: Learning rotationally equivariant features in volumetric data. In *Advances in Neural Information Processing Systems 31: Annual Conference on Neural Information Processing Systems 2018, NeurIPS*, pages 10402–10413, 2018. 2

[32] R. Winter, M. Bertolini, T. Le, F. Noé, and D. Clevert. Unsupervised learning of group invariant and equivariant representations. In *Advances in Neural Information Processing Systems 35: Annual Conference on Neural Information Processing Systems 2022, NeurIPS*, 2022. 3, 7, 8, 9, 17

[33] T. Yang, X. Ren, Y. Wang, W. Zeng, and N. Zheng. Towards building A group-based unsupervised representation disentanglement framework. In *The Tenth International Conference on Learning Representations, ICLR*, 2022. 1

[34] J. Yim, B. L. Trippe, V. D. Bortoli, E. Mathieu, A. Doucet, R. Barzilay, and T. S. Jaakkola. SE(3) diffusion model with application to protein backbone generation. In *International Conference on Machine Learning, ICML*, volume 202, pages 40001–40039, 2023. 1, 2

[35] A. Yu, V. Ye, M. Tancik, and A. Kanazawa. pixelnerf: Neural radiance fields from one or few images. In *Proceedings of the IEEE/CVF Conference on Computer Vision and Pattern Recognition*, pages 4578–4587, 2021. 3

[36] R. Zhang, P. Isola, A. A. Efros, E. Shechtman, and O. Wang. The unreasonable effectiveness of deep features as a perceptual metric. In *Proceedings of the IEEE conference on computer vision and pattern recognition*, pages 586–595, 2018. 4

# A  Model Architectures

## A.1  MNIST derived datasets

Table 4: The architecture of the encoder on two MNIST derived datasets

| |
| --- |
| Downsampling module (32 channels) |
| Downsampling module (64 channels) |
| Downsampling module (64 channels) |
| Downsampling module (64 channels) |
| Flattening |
| Fully connected (256 neurons) |
| ELU activation |
| Fully connected (24 neurons) |

Table 5: The architecture of the downsampling module on two MNIST derived datasets

| |
| --- |
| $3 \times 3$ convolution with stride 2 |
| ELU activation |

Table 6: The architecture of the decoder on two MNIST derived datasets

| |
| --- |
| Fully connected (256 neurons) |
| ELU activation |
| Fully connected (256 neurons) |
| ELU activation |
| Unflattening |
| Upsampling module (64 channels) |
| Upsampling module (64 channels) |
| Upsampling module (32 channels) |
| Upsampling module (1 channel) |

Table 7: The architecture of the upsampling module on two MNIST derived datasets

| |
| --- |
| $3 \times 3$ transposed convolution with stride 2 |
| ELU activation (Sigmoid activation for the last module) |

## A.2 Brain MRI dataset

Table 8: The architecture of the encoder on the brain MRI dataset

| |
|---|
| Downsampling module (32 channels) |
| Downsampling module (64 channels) |
| Downsampling module (64 channels) |
| 2D batch normalization |
| Downsampling module (64 channels) |
| Downsampling module (64 channels) |
| $3 \times 3$ convolution (64 channels) |
| Leaky ReLU activation |
| $3 \times 3$ convolution (64 channels) |

Table 9: The architecture of the downsampling module on the brain MRI dataset

| |
|---|
| $3 \times 3$ convolution with stride 2 |
| Leaky ReLU activation |

Table 10: The architecture of the decoder on the brain MRI dataset

| |
|---|
| $3 \times 3$ convolution (64 channels) |
| Leaky ReLU activation |
| $3 \times 3$ convolution (64 channels) |
| Leaky ReLU activation |
| Upsampling module (64 channels) |
| Upsampling module (64 channels) |
| 2D batch normalization |
| Upsampling module (64 channels) |
| Upsampling module (32 channels) |
| Upsampling module (1 channel) |

Table 11: The architecture of the upsampling module on the brain MRI dataset

| |
|---|
| $3 \times 3$ transposed convolution with stride 2 |
| ELU activation (Sigmoid activation for the last module) |

### A.3 3D objects rendered datasets

For these datasets, we employ skip connections and attention modules. For the attention model, we use the standard architecture from Vaswani et al. [29], which is not listed here. The only difference is that instead of self-attention, we have the input from the encoder acting as keys and values while the input from the decoder acting as queries. For a more intuitive illustration, see Figure 2.

Table 12: The architecture of the encoder on two 3D objects rendered datasets

Convolutional block (64 channels)
Downsampling module (128 channels)
Downsampling module (256 channels)
*Skip connection, input to attention module*
Downsampling module (512 channels)
Downsampling module (1020 channels)

Table 13: The architecture of the downsampling module on two 3D objects rendered datasets

$2 \times 2$ max pooling
Convolutional block

Table 14: The architecture of the convolutional block on two 3D objects rendered datasets

$3 \times 3$ convolution
2D batch normalization
ReLU activation
$3 \times 3$ convolution
2D batch normalization
ReLU activation

Table 15: The architecture of the decoder on two 3D objects rendered datasets

Upsampling module (512 channels)
Upsampling module (256 channels) *with skip connection*
Upsampling module (128 channels)
Upsampling module (64 channels)
Channel-wise fully connected
Sigmoid activation

Table 16: The architecture of the upsampling module (without skip connection) on two 3D objects rendered datasets

$2 \times 2$ nearest upsampling
$3 \times 3$ convolution
2D batch normalization
ReLU activation
Convolutional block

Table 17: The architecture of the upsampling module (with skip connection) on two 3D objects rendered datasets

| |
|---|
| $2 \times 2$ nearest upsampling |
| $3 \times 3$ convolution |
| 2D batch normalization |
| ReLU activation |
| *skip connection, input to attention module* |
| *attention module output, concatenate* |
| Convolutional block |

## B  Number of Learnable Parameters

We also compare the numbers of learnable parameters across our models and baselines in Table 18. As we can see, our model is relatively small which indicates we did not achieve high performance by simply using complex architectures. The only smaller baseline model is Dupont et al. [9] on the 3D rendered datasets. However, it is not any more efficient to train because of the rotation transformations on the latent coordinates requires resampling.

Table 18: Comparison of learnable parameter numbers

| | MNIST derived datasets | Brain MRI dataset | 3D rendered datasets |
|---|---|---|---|
| Winter et al. [32] | 1.22M | – | – |
| Hwang et al. [13] | 3.06M | 9.33M | – |
| Dupont et al. [9] | – | – | 11.2M |
| Sajjadi et al. [25] | – | – | 73.9M |
| **Ours** | 0.33M | 0.41M | 32.3M |

## C  Training details

Our architecture was trained on 1 A100 GPU with a batch-size of 256 using the Adam optimizer. The learning rate is 0.0001. We randomly split $1/8$ of the training set as the validation set. All models are selected based on the best validation loss. The architecture and the training was implemented in PyTorch and the code will be made available upon publication.

For the training on the rotated MNIST dataset, we added the additional loss of $\|z - E(D(z))\|_2^2$ with a weight of 0.0001 to enforce the assumption of Proposition 4.3 that the decoder is consistent with the group action. While this loss term does not improve or worsen the reconstruction performance on the test set, it induces a group action on the data space (theoretically, if it were to be exactly zero). Exploring the implications of ensuring a group action in the data space, rather than only in the latent space, is left for future work.

